# Agnostic Selective Classification

**Ran El-Yaniv  and  Yair Wiener**
Computer Science Department
Technion – Israel Institute of Technology
`{rani,wyair}@{cs,tx}.technion.ac.il`

## Abstract

For a learning problem whose associated excess loss class is $(\beta, B)$-Bernstein, we show that it is theoretically possible to track the same classification performance of the best (unknown) hypothesis in our class, provided that we are free to abstain from prediction in some region of our choice. The (probabilistic) volume of this rejected region of the domain is shown to be diminishing at rate $O(B\theta(\sqrt{1/m})^{\beta})$, where $\theta$ is Hanneke's disagreement coefficient. The strategy achieving this performance has computational barriers because it requires empirical error minimization in an agnostic setting. Nevertheless, we heuristically approximate this strategy and develop a novel selective classification algorithm using constrained SVMs. We show empirically that the resulting algorithm consistently outperforms the traditional rejection mechanism based on distance from decision boundary.

## 1 Introduction

Is it possible to achieve the same test performance as the best classifier in hindsight? The answer to this question is "probably not." However, when changing the rules of the standard game it is possible. Indeed, consider a game where our classifier is allowed to abstain from prediction, without penalty, in some region of our choice. For this case, and assuming a noise free "realizable" setting, it was shown in [1] that there is a "perfect classifier." This means that after observing only a finite labeled training sample, the learning algorithm outputs a classifier that, with certainty, will never err on any test point. To achieve this, this classifier must refuse to classify in some region of the domain. Perhaps surprisingly it was shown that the volume of this rejection region is bounded, and in fact, this volume diminishes with increasing training set sizes (under certain conditions). An open question, posed in [1], is what would be an analogous notion of perfection in an agnostic, noisy setting. Is it possible to achieve any kind of perfection in a real world scenario?

The setting under consideration, where classifiers can abstain from prediction, is called *classification with a reject option* [2, 3], or *selective classification* [1]. Focusing on this model, in this paper we present a blend of theoretical and practical results. We first show that the concept of "perfect classification" that was introduced for the realizable case in [1], can be extended to the agnostic setting. While pure perfection is impossible to accomplish in a noisy environment, a more realistic objective is to perform as well as the best hypothesis in the class within a region of our choice.

We call this type of learning "weakly optimal" selective classification and show that a novel strategy accomplishes this type of learning with diminishing rejection rate under certain Bernstein type conditions (a stronger notion of optimality is mentioned later as well). This strategy relies on empirical risk minimization, which is computationally difficult. In the practical part of the paper we present a heuristic approximation algorithm, which relies on constrained SVMs, and mimics the optimal behavior. We conclude with numerical examples that examine the empirical performance of the new algorithm and compare its performance with that of the widely used selective classification method for rejection, based on distance from decision boundary.

## 2 Selective classification and other definitions

Consider a standard agnostic binary classification setting where $\mathcal{X}$ is some feature space, and $\mathcal{H}$ is our hypothesis class of binary classifiers, $h : \mathcal{X} \rightarrow \{\pm 1\}$. Given a finite training sample of $m$ labeled examples, $S_m = \{(x_i, y_i)\}_{i=1}^m$, assumed to be sampled i.i.d. from some *unknown* underlying distribution $P(X, Y)$ over $\mathcal{X} \times \{\pm 1\}$, our goal is to select the best possible classifier from $\mathcal{H}$. For any $h \in \mathcal{H}$, its *true error*, $R(h)$, and its *empirical error*, $\hat{R}(h)$, are,

$$R(h) \triangleq \Pr_{(X,Y) \sim P} \{h(X) \neq Y\}, \quad \hat{R}(h) \triangleq \frac{1}{m} \sum_{i=1}^m \mathbb{I}\left(h(x_i) \neq y_i\right).$$

Let $\hat{h} \triangleq \arg\inf_{h \in \mathcal{H}} \hat{R}(h)$ be the *empirical risk minimizer (ERM)*, and $h^* \triangleq \arg\inf_{h \in \mathcal{H}} R(h)$, the *true risk minimizer*.

In *selective classification* [1], given $S_m$ we need to select a binary *selective classifier* defined to be a pair $(h, g)$, with $h \in \mathcal{H}$ being a standard binary classifier, and $g : \mathcal{X} \rightarrow \{0, 1\}$ is a *selection function* defining the sub-region of activity of $h$ in $\mathcal{X}$. For any $x \in \mathcal{X}$,

$$(h, g)(x) \triangleq \begin{cases} reject, & \text{if } g(x) = 0; \\ h(x), & \text{if } g(x) = 1. \end{cases} \tag{1}$$

Selective classification performance is characterized in terms of two quantities: *coverage* and *risk*. The *coverage* of $(h, g)$ is

$$\Phi(h, g) \triangleq \mathbb{E}\left[g(X)\right].$$

For a bounded loss function $\ell : \mathcal{Y} \times \mathcal{Y} \rightarrow [0, 1]$, the risk of $(h, g)$ is defined as the average loss on the accepted samples,

$$R(h, g) \triangleq \frac{\mathbb{E}\left[\ell(h(X), Y) \cdot g(X)\right]}{\Phi(h, g)}.$$

As pointed out in [1], the trade-off between risk and coverage is the main characteristic of a selective classifier. This trade-off is termed there the "risk-coverage curve" (RC curve)[1]

Let $G \subseteq \mathcal{H}$. The *disagreement set* [4, 1] w.r.t. $G$ is defined as

$$DIS(G) \triangleq \{x \in \mathcal{X} : \exists h_1, h_2 \in G \quad \text{s.t.} \quad h_1(x) \neq h_2(x)\}.$$

For any hypothesis class $\mathcal{H}$, target hypothesis $h \in \mathcal{H}$, distribution $P$, sample $S_m$, and real $r > 0$, define

$$\mathcal{V}(h, r) = \{h' \in \mathcal{H} : R(h') \leq R(h) + r\} \quad \text{and} \quad \hat{\mathcal{V}}(h, r) = \left\{h' \in \mathcal{H} : \hat{R}(h') \leq \hat{R}(h) + r\right\}. \tag{2}$$

Finally, for any $h \in \mathcal{H}$ we define a ball in $\mathcal{H}$ of radius $r$ around $h$ [5]. Specifically, with respect to class $\mathcal{H}$, marginal distribution $P$ over $\mathcal{X}$, $h \in \mathcal{H}$, and real $r > 0$, define

$$\mathcal{B}(h, r) \triangleq \left\{h' \in \mathcal{H} : \Pr_{X \sim P} \{h'(X) \neq h(X)\} \leq r\right\}.$$

## 3 Perfect and weakly optimal selective classifiers

The concept of *perfect classification* was introduced in [1] within a *realizable* selective classification setting. Perfect classification is an extreme case of selective classification where a selective classifier $(h, g)$ achieves $R(h, g) = 0$ with certainty; that is, the classifier never errs on its region of activity. Obviously, the classifier must compromise sufficiently large part of the domain $\mathcal{X}$ in order to achieve this outstanding performance. Surprisingly, it was shown in [1] that not-trivial perfect classification exists in the sense that under certain conditions (e.g., finite hypothesis class) the rejected region diminishes at rate $\Omega(1/m)$, where $m$ is the size of the training set.

In agnostic environments, as we consider here, such perfect classification appears to be out of reach. In general, in the worst case no hypothesis can achieve zero error over any nonempty subset of the

domain. We consider here the following weaker, but still extremely desirable behavior, which we call "weakly optimal selective classification." Let $h^* \in H$ be the true risk minimizer of our problem. Let $(h, g)$ be a selective classifier selected after observing the training set $S_m$. We say that $(h, g)$ is a *weakly optimal* selective classifier if, for any $0 < \delta < 1$, with probability of at least $1 - \delta$ over random choices of $S_m$, $R(h, g) \leq R(h^*, g)$. That is, with high probability our classifier is at least as good as the true risk minimizer over its region of activity. We call this classifier 'weakly optimal' because a stronger requirement would be that the classifier should achieve the best possible error among all hypotheses in $\mathcal{H}$ restricted to the region of activity defined by $g$.

## 4 A learning strategy

We now present a strategy that will be shown later to achieve non-trivial weakly optimal selective classification under certain conditions. We call it a "strategy" rather than an "algorithm" because it does not include implementation details.

Let's begin with some motivation. Using standard concentration inequalities one can show that the training error of the true risk minimizer, $h^*$, cannot be "too far" from the training error of the empirical risk minimizer, $\hat{h}$. Therefore, we can guarantee, with high probability, that the class of all hypothesis with "sufficiently low" empirical error includes the true risk minimizer $h^*$. Selecting only subset of the domain, for which all hypothesis in that class agree, is then sufficient to guarantee weak optimality. Strategy 1 formulates this idea. In the next section we analyze this strategy and show that it achieves this optimality with non trivial (bounded) coverage.

---

**Strategy 1** Learning strategy for weakly optimal selective classifiers

---
**Input:** $S_m, m, \delta, d$
**Output:** a selective classifier $(h, g)$ such that $R(h, g) = R(h^*, g)$ w.p. $1 - \delta$
  1: Set $\hat{h} = ERM(\mathcal{H}, S_m)$, i.e., $\hat{h}$ is any empirical risk minimizer from $\mathcal{H}$
  2: Set $G = \hat{\mathcal{V}}\left(\hat{h}, 4\sqrt{2\frac{d\left(\ln \frac{2me}{d}\right) + \ln \frac{8}{\delta}}{m}}\right)$ (see Eq. (2))
  3: Construct $g$ such that $g(x) = 1 \iff x \in \{\mathcal{X} \setminus DIS(G)\}$
  4: $h = \hat{h}$

---

## 5 Analysis

We begin with a few definitions. Consider an instance of a binary learning problem with hypothesis class $\mathcal{H}$, an underlying distribution $P$ over $\mathcal{X} \times \mathcal{Y}$, and a loss function $\ell(\mathcal{Y}, \mathcal{Y})$. Let $h^* = \arg\inf_{h \in \mathcal{H}} \{\mathbb{E}\ell(h(X), Y)\}$ be the true risk minimizer. The associated *excess loss class* [6] is defined as
$$\mathcal{F} \triangleq \{\ell(h(x), y) - \ell(h^*(x), y) : h \in \mathcal{H}\}.$$
Class $\mathcal{F}$ is said to be a $(\beta, B)$-*Bernstein* class with respect to $P$ (where $0 < \beta \leq 1$ and $B \geq 1$), if every $f \in \mathcal{F}$ satisfies
$$\mathbb{E}f^2 \leq B(\mathbb{E}f)^\beta.$$
Bernstein classes arise in many natural situations; see discussions in [7, 8]. For example, if the probability $P(X, Y)$ satisfies Tsybakov's noise conditions then the excess loss function is a Bernstein [8, 9] class. In the following sequence of lemmas and theorems we assume a binary hypothesis class $\mathcal{H}$ with VC-dimension $d$, an underlying distribution $P$ over $\mathcal{X} \times \{\pm 1\}$, and $\ell$ is the 0/1 loss function. Also, $\mathcal{F}$ denotes the associated excess loss class. Our results can be extended to losses other than 0/1 by similar techniques to those used in [10].

**Lemma 5.1.** *If $\mathcal{F}$ is a $(\beta, B)$-Bernstein class with respect to $P$, then for any $r > 0$*
$$\mathcal{V}(h^*, r) \subseteq \mathcal{B}\left(h^*, Br^\beta\right).$$

*Proof.* If $h \in \mathcal{V}(h^*, r)$ then, by definition
$$\mathbb{E}\{\mathbb{I}(h(X) \neq Y)\} \leq \mathbb{E}\{\mathbb{I}(h^*(X) \neq Y)\} + r.$$

Using the linearity of expectation we have,

$$\mathbb{E}\left\{\mathbb{I}(h(X) \neq Y) - \mathbb{I}(h^*(X) \neq Y)\right\} \leq r. \tag{3}$$

Since $\mathcal{F}$ is a $(\beta, B)$-Bernstein class,

$$
\begin{aligned}
\mathbb{E}\left\{\mathbb{I}(h(X) \neq h^*(X))\right\} &= \mathbb{E}\left\{|\mathbb{I}(h(X) \neq Y) - \mathbb{I}(h^*(X) \neq Y)|\right\} \\
&= \mathbb{E}\left\{(\ell(h(X), Y) - \ell(h^*(X), Y))^2\right\} = \mathbb{E}f^2 \leq B(\mathbb{E}f)^\beta \\
&= B\left(\mathbb{E}\left\{\mathbb{I}(h(X) \neq Y) - \mathbb{I}(h^*(X) \neq Y)\right\}\right)^\beta.
\end{aligned}
$$

By (3), for any $r > 0$, $\mathbb{E}\left\{\mathbb{I}(h(X) \neq h^*(X))\right\} \leq Br^\beta$. Therefore, by definition, $h \in \mathcal{B}\left(h^*, Br^\beta\right)$.
$\square$

Throughout this section we denote

$$\sigma(m, \delta, d) \triangleq 2\sqrt{2\frac{d\left(\ln\frac{2me}{d}\right) + \ln\frac{2}{\delta}}{m}}.$$

**Theorem 5.2** ([11])**.** *For any $0 < \delta < 1$, with probability of at least $1 - \delta$ over the choice of $S_m$ from $P^m$, any hypothesis $h \in \mathcal{H}$ satisfies*

$$R(h) \leq \hat{R}(h) + \sigma(m, \delta, d).$$

*Similarly $\hat{R}(h) \leq R(h) + \sigma(m, \delta, d)$ under the same conditions.*

**Lemma 5.3.** *For any $r > 0$, and $0 < \delta < 1$, with probability of at least $1 - \delta$,*

$$\hat{\mathcal{V}}(\hat{h}, r) \subseteq \mathcal{V}\left(h^*, 2\sigma(m, \delta/2, d) + r\right).$$

*Proof.* If $h \in \hat{\mathcal{V}}(\hat{h}, r)$, then, by definition, $\hat{R}(h) \leq \hat{R}(\hat{h}) + r$. Since $\hat{h}$ minimizes the empirical error, we have, $\hat{R}(\hat{h}) \leq \hat{R}(h^*)$. Using Theorem 5.2 twice, and applying the union bound, we know that w.p. of at least $1 - \delta$,

$$R(h) \leq \hat{R}(h) + \sigma(m, \delta/2, d) \quad \wedge \quad \hat{R}(h^*) \leq R(h^*) + \sigma(m, \delta/2, d).$$

Therefore, $R(h) \leq R(h^*) + 2\sigma(m, \delta/2, d) + r$, and $h \in \mathcal{V}\left(h^*, 2\sigma(m, \delta/2, d) + r\right)$. $\square$

For any $G \subseteq \mathcal{H}$, and distribution $P$ we define, $\Delta G \triangleq \Pr\{DIS(G)\}$. Hanneke introduced a complexity measure for active learning problems termed the *disagreement coefficient* [5]. The disagreement coefficient of $h$ with respect to $\mathcal{H}$ under distribution $P$ is,

$$\theta_h \triangleq \sup_{r > \epsilon} \frac{\Delta \mathcal{B}(h, r)}{r}, \tag{4}$$

where $\epsilon = 0$. The disagreement coefficient of the hypothesis class $\mathcal{H}$ with respect to $P$ is defined as

$$\theta \triangleq \limsup_{k \to \infty} \theta_{h^{(k)}},$$

where $\left\{h^{(k)}\right\}$ is any sequence of $h^{(k)} \in \mathcal{H}$ with $R(h^{(k)})$ monotonically decreasing.

**Theorem 5.4.** *Assume that $\mathcal{H}$ has disagreement coefficient $\theta$ and that $\mathcal{F}$ is a $(\beta, B)$-Bernstein class w.r.t. $P$. Then, for any $r > 0$ and $0 < \delta < 1$, with probability of at least $1 - \delta$,*

$$\Delta \hat{\mathcal{V}}(\hat{h}, r) \leq B\theta\left(2\sigma(m, \delta/2, d) + r\right)^\beta.$$

*Proof.* Applying Lemmas 5.3 and 5.1 we get that with probability of at least $1 - \delta$,

$$\hat{\mathcal{V}}(\hat{h}, r) \subseteq \mathcal{B}\left(h^*, B\left(2\sigma(m, \delta/2, d) + r\right)^\beta\right).$$

Therefore

$$\Delta \hat{\mathcal{V}}(\hat{h}, r) \leq \Delta \mathcal{B}\left(h^*, B\left(2\sigma(m, \delta/2, d) + r\right)^\beta\right).$$

By the definition of the disagreement coefficient, for any $r' > 0$, $\Delta \mathcal{B}(h^*, r') \leq \theta r'$. $\square$

**Theorem 5.5.** *Assume that $\mathcal{H}$ has disagreement coefficient $\theta$ and that $\mathcal{F}$ is a $(\beta, B)$-Bernstein class w.r.t. $P$. Let $(h, g)$ be the selective classifier chosen by Algorithm 1. Then, with probability of at least $1 - \delta$,*

$$\Phi(h, g) \geq 1 - B\theta \left(4\sigma(m, \delta/4, d)\right)^{\beta} \qquad \wedge \qquad R(h, g) = R(h^*, g).$$

*Proof.* Applying Theorem 5.2 we get that with probability of at least $1 - \delta/4$,

$$\hat{R}(h^*) \leq R(h^*) + \sigma(m, \delta/4, d).$$

Since $h^*$ minimizes the true error, wet get that $R(h^*) \leq R(\hat{h})$. Applying again Theorem 5.2 we know that with probability of at least $1 - \delta/4$, $R(\hat{h}) \leq \hat{R}(\hat{h}) + \sigma(m, \delta/4, d)$. Applying the union bound we have that with probability of at least $1 - \delta/2$, $\hat{R}(h^*) \leq \hat{R}(\hat{h}) + 2\sigma(m, \delta/4, d)$. Hence, with probability of at least $1 - \delta/2$, $h^* \in \hat{\mathcal{V}}\left(\hat{h}, 2\sigma(m, \delta/4, d)\right) = G$. We note that the selection function $g(x)$ equals one only for $x \in \mathcal{X} \setminus DIS(G)$. Therefore, for any $x \in \mathcal{X}$, for which $g(x) = 1$, all the hypotheses in $G$ agree, and in particular $h^*$ and $\hat{h}$ agree. Thus,

$$R(\hat{h}, g) \quad = \quad \frac{\mathbb{E}\{\mathbb{I}(\hat{h}(X) \neq Y) \cdot g(X)\}}{\mathbb{E}\{g(X)\}} = \frac{\mathbb{E}\left\{\mathbb{I}(h^*(X) \neq Y) \cdot g(X)\right\}}{\mathbb{E}\{g(X)\}} = R(h^*, g).$$

Applying Theorem 5.4 and the union bound we therefore know that with probability of at least $1 - \delta$,

$$\Phi(\hat{h}, g) = \mathbb{E}\{g(X)\} = 1 - \Delta G \geq 1 - B\theta \left(4\sigma(m, \delta/4, d)\right)^{\beta}.$$

$\square$

Hanneke introduced, in his original work [5], an alternative definition of the disagreement coefficient $\theta$, for which the supremum in (4) is taken with respect to any fixed $\epsilon > 0$. Using this alternative definition it is possible to show that fast coverage rates are achievable, not only for finite disagreement coefficients (Theorem 5.5), but also if the disagreement coefficient grows slowly with respect to $1/\epsilon$ (as shown by Wang [12], under sufficient smoothness conditions). This extension will be discussed in the full version of this paper.

# 6   A disbelief principle and the risk-coverage trade-off

Theorem 5.5 tells us that the strategy presented in Section 4 not only outputs a weakly optimal selective classifier, but this classifier also has guaranteed coverage (under some conditions). As emphasized in [1], in practical applications it is desirable to allow for some control on the trade-off between risk and coverage; in other words, we would like to be able to develop the entire risk-coverage curve for the classifier at hand and select ourselves the cutoff point along this curve in accordance with other practical considerations we may have. How can this be achieved?

The following lemma facilitates a construction of a risk-coverage trade-off curve. The result is an alternative characterization of the selection function $g$, of the weakly optimal selective classifier chosen by Strategy 1. This result allows for calculating the value of $g(x)$, for any individual test point $x \in \mathcal{X}$, without actually constructing $g$ for the entire domain $\mathcal{X}$.

**Lemma 6.1.** *Let $(h, g)$ be a selective classifier chosen by Strategy 1 after observing the training sample $S_m$. Let $\hat{h}$ be the empirical risk minimizer over $S_m$. Let $x$ be any point in $\mathcal{X}$ and*

$$\widetilde{h}_x \triangleq \operatorname*{argmin}_{h \in \mathcal{H}} \left\{ \hat{R}(h) \quad | \quad h(x) = -sign\left(\hat{h}(x)\right) \right\},$$

*an empirical risk minimizer forced to label $x$ the opposite from $\hat{h}(x)$. Then*

$$g(x) = 0 \quad \Longleftrightarrow \quad \hat{R}(\widetilde{h}_x) - \hat{R}(\hat{h}) \leq 2\sigma(m, \delta/4, d).$$

*Proof.* According to the definition of $\hat{\mathcal{V}}$ (see Eq. (2)),

$$\hat{R}(\widetilde{h}_x) - \hat{R}(\hat{h}) \leq 2\sigma(m, \delta/4, d) \quad \Longleftrightarrow \quad \widetilde{h} \in \hat{\mathcal{V}}\left(\hat{h}, 2\sigma(m, \delta/4, d)\right)$$

Thus, $\hat{h}, \widetilde{h}_x \in \hat{\mathcal{V}}$. However, by construction, $\hat{h}(x) = -\widetilde{h}(x)$, so $x \in DIS(\hat{\mathcal{V}})$ and $g(x) = 0$. $\square$

Lemma 6.1 tells us that in order to decide if point $x$ should be rejected we need to measure the empirical error $\hat{R}(\widetilde{h}_x)$ of a special empirical risk minimizer, $\widetilde{h}_x$, which is constrained to label $x$ the opposite from $\hat{h}(x)$. If this error is sufficiently close to $\hat{R}(\hat{h})$ our classifier cannot be too sure about the label of $x$ and we must reject it. This result strongly motivates the following definition of a "disbelief index" for each individual point.

**Definition 6.2 (disbelief index).** *For any $x \in \mathcal{X}$, define its disbelief index w.r.t. $S_m$ and $\mathcal{H}$,*

$$D(x) \triangleq D(x, S_m) \triangleq \hat{R}(\widetilde{h}_x) - \hat{R}(\hat{h}).$$

Observe that $D(x)$ is large whenever our model is sensitive to label of $x$ in the sense that when we are forced to bend our best model to fit the opposite label of $x$, our model substantially deteriorates, giving rise to a large disbelief index. This large $D(x)$ can be interpreted as our disbelief in the possibility that $x$ can be labeled so differently. In this case we should definitely predict the label of $x$ using our unforced model. Conversely, if $D(x)$ is small, our model is indifferent to the label of $x$ and in this sense, is not committed to its label. In this case we should abstain from prediction at $x$.

This "disbelief principle" facilitates an exploration of the risk-coverage trade-off curve for our classifier. Given a pool of test points we can rank these test points according to their disbelief index, and points with low index should be rejected first. Thus, this ranking provides the means for constructing a risk-coverage trade-off curve.

A similar technique of using an ERM oracle that can enforce an arbitrary number of example-based constraints was used in [13, 14] in the context of active learning. As in our disbelief index, the difference between the empirical risk (or importance weighted empirical risk [14]) of two ERM oracles (with different constraints) is used to estimate prediction confidence.

# 7 Implementation

At this point in the paper we switch from theory to practice, aiming at implementing rejection methods inspired by the disbelief principle and see how well they work on real world (well, ..., UCI) problems. Attempting to implement a learning algorithm driven by the disbelief index we face a major bottleneck because the calculation of the index requires the identification of ERM hypotheses. To handle this computationally difficult problem, we "approximate" the ERM as follows. Focusing on SVMs we use a high $C$ value ($10^5$ in our experiments) to penalize more on training errors than on small margin. In this way the solution to the optimization problem tend to get closer to the ERM.

Another problem we face is that the disbelief index is a noisy statistic that highly depends on the sample $S_m$. To overcome this noise we use robust statistics. First we generate 11 different samples $(S_m^1, S_m^2, \dots S_m^{11})$ using bootstrap sampling. For each sample we calculate the disbelief index for all test points and for each point take the median of these measurements as the final index.

We note that for any finite training sample the disbelief index is a discrete variable. It is often the case that several test points share the same disbelief index. In those cases we can use any confidence measure as a tie breaker. In our experiments we use distance from decision boundary to break ties.

In order to estimate $\hat{R}(\widetilde{h}_x)$ we have to restrict the SVM optimizer to only consider hypotheses that classify the point $x$ in a specific way. To accomplish this we use a weighted SVM for unbalanced data. We add the point $x$ as another training point with weight 10 times larger than the weight of all training points combined. Thus, the penalty for misclassification of $x$ is very large and the optimizer finds a solution that doesn't violate the constraint.

# 8 Empirical results

Focusing on SVMs with a linear kernel we compared the RC (Risk-Coverage) curves achieved by the proposed method with those achieved by SVM with rejection based on distance from decision boundary. This latter approach is very common in practical applications of selective classification. For implementation we used LIBSVM [15].

Before presenting these results we wish to emphasize that the proposed method leads to rejection regions fundamentally different than those obtained by the traditional distance-based technique. In

Figure 1 we depict those regions for a training sample of 150 points sampled from a mixture of two identical normal distributions (centered at different locations). The height map reflects the "confidence regions" of each technique according to its own confidence measure.

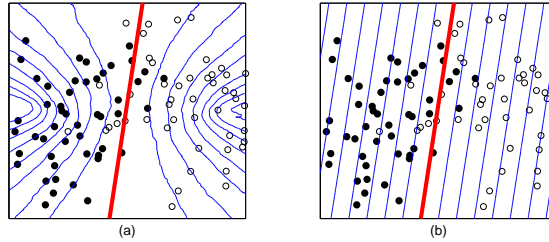

Figure 1: confidence height map using (a) disbelief index; (b) distance from decision boundary.

We tested our algorithm on standard medical diagnosis problems from the UCI repository, including all datasets used in [16]. We transformed nominal features to numerical ones in a standard way using binary indicator attributes. We also normalized each attribute independently so that its dynamic range is $[0, 1]$. No other preprocessing was employed.

In each iteration we choose uniformly at random non overlapping training set (100 samples) and test set (200 samples) for each dataset. SVM was trained on the entire training set and test samples were sorted according to confidence (either using distance from decision boundary or disbelief index). Figure 2 depicts the RC curves of our technique (red solid line) and rejection based on distance from decision boundary (green dashed line) for linear kernel on all 6 datasets. All results are averaged over 500 iterations (error bars show standard error).

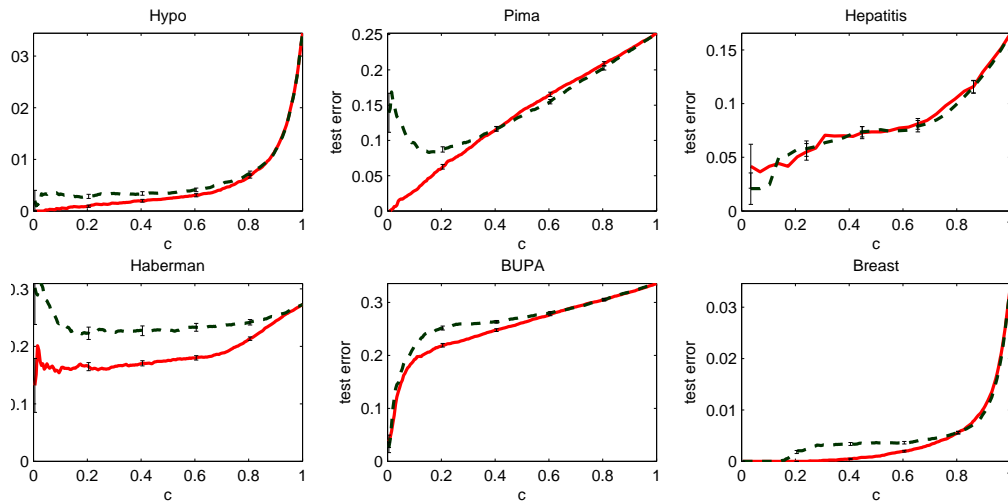

Figure 2: RC curves for SVM with linear kernel. Our method in solid red, and rejection based on distance from decision boundary in dashed green. Horizntal axis (c) represents coverage.

With the exception of the Hepatitis dataset, in which both methods were statistically indistinguishable, in all other datasets the proposed method exhibits significant advantage over the traditional approach. We would like to highlight the performance of the proposed method on the Pima dataset. While the traditional approach cannot achieve error less than $8\%$ for any rejection rate, in our approach the test error decreases monotonically to zero with rejection rate. Furthermore, a clear advantage for our method over a large range of rejection rates is evident in the Haberman dataset.[2]

For the sake of fairness, we note that the running time of our algorithm (as presented here) is substantially longer than the traditional technique. The performance of our algorithm can be substantially improved when many unlabeled samples are available. Details will be provided in the full paper.

# 9    Related work

The literature on theoretical studies of selective classification is rather sparse. El-Yaniv and Wiener [1] studied the performance of a simple selective learning strategy for the realizable case. Given an hypothesis class $\mathcal{H}$, and a sample $S_m$, their method abstain from prediction if all hypotheses in the version space do not agree on the target sample. They were able to show that their selective classifier achieves perfect classification with meaningful coverage under some conditions. Our work can be viewed as an extension of the above algorithm to the agnostic case.

Freund et al. [18] studied another simple ensemble method for binary classification. Given an hypothesis class $\mathcal{H}$, the method outputs a weighted average of all the hypotheses in $\mathcal{H}$, where the weight of each hypothesis exponentially depends on its individual training error. Their algorithm abstains from prediction whenever the weighted average of all individual predictions is close to zero. They were able to bound the probability of misclassification by $2R(h^*) + \epsilon(m)$ and, under some conditions, they proved a bound of $5R(h^*) + \epsilon(\mathcal{F}, m)$ on the rejection rate. Our algorithm can be viewed as an extreme variation of the Freund et al. method. We include in our "ensemble" only hypotheses with sufficiently low empirical error and we abstain if the weighted average of all predictions is not definitive ($\neq \pm 1$). Our risk and coverage bounds are asymptotically tighter.

Excess risk bounds were developed by Herbei and Wegkamp [19] for a model where each rejection incurs a cost $0 \le d \le 1/2$. Their bound applies to any empirical risk minimizer over a hypothesis class of ternary hypotheses (whose output is in $\{\pm 1, \text{reject}\}$). See also various extensions [20, 21].

A rejection mechanism for SVMs based on distance from decision boundary is perhaps the most widely known and used rejection technique. It is routinely used in medical applications [22, 23, 24]. Few papers proposed alternative techniques for rejection in the case of SVMs. Those include taking the reject area into account during optimization [25], training two SVM classifiers with asymmetric cost [26], and using a hinge loss [20]. Grandvalet et al. [16] proposed an efficient implementation of SVM with a reject option using a double hinge loss. They empirically compared their results with two other selective classifiers: the one proposed by Bartlett and Wegkamp [20] and the traditional rejection based on distance from decision boundary. In their experiments there was no statistically significant advantage to either method compared to the traditional approach for high rejection rates.

# 10    Conclusion

We presented and analyzed a learning strategy for selective classification that achieves weak optimality. We showed that the coverage rate directly depends on the disagreement coefficient, thus linking between active learning and selective classification. Recently it has been shown that, for the noise-free case, active learning can be reduced to selective classification [27]. We conjecture that such a reduction also holds in noisy settings. Exact implementation of our strategy, or exact computation of the disbelief index may be too difficult to achieve or even obtain with approximation guarantees. We presented one algorithm that heuristically approximate the required behavior and there is certainly room for other, perhaps better methods and variants. Our empirical examination of the proposed algorithm indicate that it can provide significant and consistent advantage over the traditional rejection technique with SVMs. This advantage can be of great value especially in medical diagnosis applications and other mission critical classification tasks. The algorithm itself can be implemented using off-the-shelf packages.

**Acknowledgments**

This work was supported in part by the IST Programme of the European Community, under the PASCAL2 Network of Excellence, IST-2007-216886. This publication only reflects the authors' views.

## Footnotes

[1]Some authors refer to an equivalent variant of this curve as "Accuracy-Rejection Curve" or *ARC*.

[2] The Haberman dataset contains survival data of patients who had undergone surgery for breast cancer. With estimated 207,090 new cases of breast cancer in the united states during 2010 [17] an improvement of $1\%$ affects the lives of more than 2000 women.

# References

[1] R. El-Yaniv and Y. Wiener. On the foundations of noise-free selective classification. *JMLR*, 11:1605–1641, 2010.

[2] C.K. Chow. An optimum character recognition system using decision function. *IEEE Trans. Computer*, 6(4):247–254, 1957.

[3] C.K. Chow. On optimum recognition error and reject trade-off. *IEEE Trans. on Information Theory*, 16:41–36, 1970.

[4] S. Hanneke. A bound on the label complexity of agnostic active learning. In *ICML*, pages 353–360, 2007.

[5] S. Hanneke. *Theoretical Foundations of Active Learning*. PhD thesis, Carnegie Mellon University, 2009.

[6] P.L. Bartlett, S. Mendelson, and P. Philips. Local complexities for empirical risk minimization. In *COLT: Proceedings of the Workshop on Computational Learning Theory, Morgan Kaufmann Publishers*, 2004.

[7] V. Koltchinskii. 2004 IMS medallion lecture: Local rademacher complexities and oracle inequalities in risk minimization. *Annals of Statistics*, 34:2593–2656, 2006.

[8] P.L. Bartlett and S. Mendelson. Discussion of "2004 IMS medallion lecture: Local rademacher complexities and oracle inequalities in risk minimization" by V. koltchinskii. *Annals of Statistics*, 34:2657–2663, 2006.

[9] A.B. Tsybakov. Optimal aggregation of classifiers in statistical learning. *Annals of Mathematical Statistics*, 32:135–166, 2004.

[10] A. Beygelzimer, S. Dasgupta, and J. Langford. Importance weighted active learning. In *ICML '09: Proceedings of the 26th Annual International Conference on Machine Learning*, pages 49–56. ACM, 2009.

[11] O. Bousquet, S. Boucheron, and G. Lugosi. Introduction to statistical learning theory. In *Advanced Lectures on Machine Learning*, volume 3176 of *Lecture Notes in Computer Science*, pages 169–207. Springer, 2003.

[12] L. Wang. Smoothness, disagreement coefficient, and the label complexity of agnostic active learning. *JMLR*, pages 2269–2292, 2011.

[13] S. Dasgupta, D. Hsu, and C. Monteleoni. A general agnostic active learning algorithm. In *NIPS*, 2007.

[14] A. Beygelzimer, D. Hsu, J. Langford, and T. Zhang. Agnostic active learning without constraints. *Advances in Neural Information Processing Systems 23*, 2010.

[15] C.C. Chang and C.J. Lin. LIBSVM: A library for support vector machines. *ACM Transactions on Intelligent Systems and Technology*, 2:27:1–27:27, 2011. Software available at "http://www.csie.ntu.edu.tw/ cjlin/libsvm".

[16] Y. Grandvalet, A. Rakotomamonjy, J. Keshet, and S. Canu. Support vector machines with a reject option. In *NIPS*, pages 537–544. MIT Press, 2008.

[17] American Cancer Society. Cancer facts and figures. 2010.

[18] Y. Freund, Y. Mansour, and R.E. Schapire. Generalization bounds for averaged classifiers. *Annals of Statistics*, 32(4):1698–1722, 2004.

[19] R. Herbei and M.H. Wegkamp. Classification with reject option. *The Canadian Journal of Statistics*, 34(4):709–721, 2006.

[20] P.L. Bartlett and M.H. Wegkamp. Classification with a reject option using a hinge loss. *Journal of Machine Learning Research*, 9:1823–1840, 2008.

[21] M.H. Wegkap. Lasso type classifiers with a reject option. *Electronic Journal of Statistics*, 1:155–168, 2007.

[22] S. Mukherjee, P. Tamayo, D. Slonim, A. Verri, T. Golub, J. P. Mesirov, and T. Poggio. Support vector machine classification of microarray data. Technical report, AI Memo 1677, Massachusetts Institute of Technology, 1998.

[23] I. Guyon, J. Weston, S. Barnhill, and V. Vapnik. Gene selection for cancer classification using support vector machines. *Machine Learning*, pages 389–422, 2002.

[24] S. Mukherjee. Chapter 9. classifying microarray data using support vector machines. In *of scientists from the University of Pennsylvania School of Medicine and the School of Engineering and Applied Science*. Kluwer Academic Publishers, 2003.

[25] G. Fumera and F. Roli. Support vector machines with embedded reject option. In *Pattern Recognition with Support Vector Machines: First International Workshop*, pages 811–919, 2002.

[26] R. Sousa, B. Mora, and J.S. Cardoso. An ordinal data method for the classification with reject option. In *ICMLA*, pages 746–750. IEEE Computer Society, 2009.

[27] R. El-Yaniv and Y. Wiener. Active learning via perfect selective classification. Accepted to *JMLR*, 2011.

